# Visual gesture-based robot guidance with a modular neural system

**E. Littmann,**
Abt. Neuroinformatik, Fak. f. Informatik
Universität Ulm, D-89069 Ulm, FRG
enno@neuro.informatik.uni-ulm.de

**A. Drees, and H. Ritter**
AG Neuroinformatik, Techn. Fakultät
Univ. Bielefeld, D-33615 Bielefeld, FRG
andrea,helge@techfak.uni-bielefeld.de

## Abstract

We report on the development of the modular neural system "SEE-EAGLE" for the visual guidance of robot pick-and-place actions. Several neural networks are integrated to a single system that visually recognizes human hand pointing gestures from stereo pairs of color video images. The output of the hand recognition stage is processed by a set of color-sensitive neural networks to determine the cartesian location of the target object that is referenced by the pointing gesture. Finally, this information is used to guide a robot to grab the target object and put it at another location that can be specified by a second pointing gesture. The accuracy of the current system allows to identify the location of the referenced target object to an accuracy of 1 cm in a workspace area of 50x50 cm. In our current environment, this is sufficient to pick and place arbitrarily positioned target objects within the workspace. The system consists of neural networks that perform the tasks of image segmentation, estimation of hand location, estimation of 3D-pointing direction, object recognition, and necessary coordinate transforms. Drawing heavily on the use of learning algorithms, the functions of all network modules were created from data examples only.

## 1 Introduction

The rapidly developing technology in the fields of robotics and virtual reality requires the development of new and more powerful interfaces for configuration and control of such devices. These interfaces should be intuitive for the human advisor and comfortable to use. Practical solutions so far require the human to wear a device that can transfer the necessary information. One typical example is the data glove [14, 12]. Clearly, in the long run solutions that are contactless will be much more desirable, and vision is one of the major modalities that appears especially suited for the realization of such solutions.

In the present paper, we focus on a still restricted but very important task in robot control, the guidance of robot pick-and-place actions by unconstrained human pointing gestures in a realistic laboratory environment. The input of target locations by

pointing gestures provides a powerful, very intuitive and comfortable functionality for a vision-based man-machine interface for guiding robots and extends previous work that focused on the detection of hand location or the discrimination of a small, discrete number of hand gestures only [10, 1, 2, 8]. Besides two color cameras, no special device is necessary to evaluate the gesture of the human operator.

A second goal of our approach is to investigate how to build a neural system for such a complex task from several neural modules. The development of advanced artificial neural systems challenges us with the task of finding architectures for the cooperation of multiple functional modules such that part of the structure of the overall system can be designed at a useful level of abstraction, but at the same time learning can be used to create or fine-tune the functionality of parts of the system on the basis of suitable training examples.

To approach this goal requires to shift the focus from exploring the properties of single networks to exploring the properties of entire *systems of neural networks*. The work on "mixtures of experts" [3, 4] is one important contribution along these lines. While this is a widely applicable and powerful approach, there clearly is a need to go beyond the exploration of strictly hierarchical systems and to gain experience with architectures that admit more complex types of information flow as required e.g. by the inclusion of features such as control of focal attention or reentrant processing branches. The need for such features arose very naturally in the context of the task described above, and in the following section we will report our results with a system architecture that is crucially based on the exploitation of such elements.

## 2   System architecture

Our system, described in fig. 1, is situated in a complex laboratory environment. A robot arm with manipulator is mounted at one side of a table with several objects of different color placed on it. A human operator is positioned at the next side to the right of the robot. This scenery is watched by two cameras from the other two sides from high above. The cameras yield a stereo color image of the scene (images I0). The operator points with one hand at one of the objects on the table. On the basis of the image information, the object is located and the robot grabs it. Then, the operator points at another location, where the robot releases the object.[1]

The system consists of several hardware components: a PUMA 560 robot arm with six axes and a three-fingered manipulator [2]; two single-chip PULNIX color cameras; two ANDROX vision boards with software for data acquisition and processing; a work space consisting of a table with a black grid on a yellow surface. Robot and person refer to the same work space. Both cameras must show both the human hand and the table with the objects. Within this constraint, the position of the cameras can be chosen freely as long as they yield significantly different views.

An important prerequisite for the recognition of the pointing direction is the segmentation of the human hand from the background scenery. This task is solved by a LLM network (S1) trained to yield a probability value for each image pixel to belong to the hand region. The training is based on the local color information. This procedure has been investigated in [7].

An important feature of the chosen method is the great reliability and robustness of both the classification performance and the localization accuracy of the searched object. Furthermore, the performance is quite constant over a wide range of image resolutions. This allows a fast two-step procedure: First, the images are segmented in low resolution (S1: I1 → A1) and the hand position is extracted. Then, a small

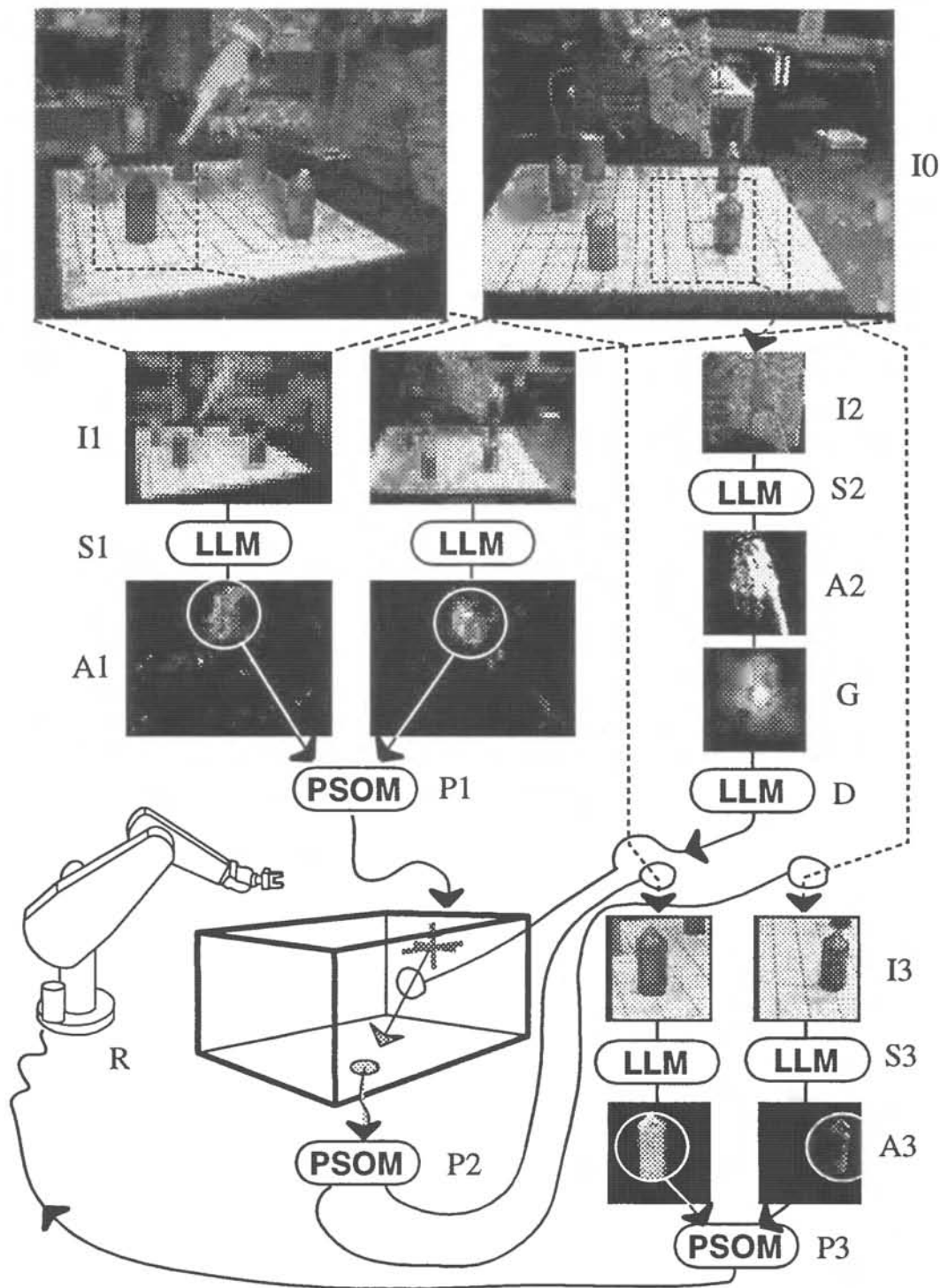

Fig. 1: System architecture. From two color camera images I0 we extract the hand position (I1 ▷ S1 ▷ A1 (pixel coord.) ▷ P1 ▷ cartesian hand coord.). In a subframe centered on the hand location (I2) we determine the pointing direction (I2 ▷ S2 ▷ A2 (pixel coord.) ▷ G ▷ D ▷ pointing angles). Pointing direction and hand location define a cartesian target location that is mapped to image coord. that define the centers of object subframes (I0 ▷ P2 ▷ I3). There we determine the target object (I3 ▷ S3 ▷ A3) and map the pixel coord. of its centers to world coord. (A3 ▷ P3 ▷ world target loc.). These coordinates are used to guide the robot R to the target object.

subframe (I2) around the estimated hand position is processed in high resolution by another dedicated LLM network (S2: I2 → A2). For details of the segmentation process, refer to [6].

The extraction of hand information by LLMs on the basis of Gabor masks has already been studied for hand posture [9] and orientation [5]. The method is based on a segmented image containing the hand only (A2). This image is filtered by 36 Gabor masks that are arranged on a 3x3 grid with 4 directions per grid position and centered on the hand. The filter kernels have a radius of 10 pixels, the distance between the grid points is 20 pixels. The 36 filter responses (G) form the input vector for a LLM network (D). Further details of the processing are reported in [6].

The network yields the pointing direction of the hand (D: I2 → G → pointing direction). Together with the hand position which is computed by a parametrized self-organizing map ("PSOM", see below and [11, 13]) (P1: A1 → cartesian hand position), a (cartesian) target location in the workspace can be calculated. This location can be retransformed by the PSOM into pixel coordinates (P2: cartesian target location → target pixel coordinates). These coordinates define the center of an "attention region" (I3) that is searched for a set of predefined target objects. This object recognition is performed by a set of LLM color segmentation networks (S3: I3 → A3), each previously trained for one of the defined targets. A ranking procedure is used to determine the target object. The pixel coordinates of the target in the segmented image are mapped by the PSOM to world coordinates (P3: A3 → cartesian target position). The robot R now moves to above these world coordinates, moves vertically down, grabs whatever is there, and moves upward again. Now, the system evaluates a second pointing gesture that specifies the place where to place the object. This time, the world coordinates calculated on the basis of the pointing direction from network D and the cartesian hand location from PSOM P1 serve directly as target location for the robot.

For our processing we must map corresponding pixels in the stereo images to cartesian world coordinates. For these transformations, training data was generated with aid of the robot on a precise sampling grid. We automatically extract the pixel coordinates of a LED at the tip of the robot manipulator from both images. The seven-dimensional feature vector serves as training input for an PSOM network [11]. By virtue of its capability to represent a transformation in a symmetric, "multiway"-fashion, this offers the additional benefit that both the camera-to-world mapping *and* its inverse can be obtained with a single network trained only once on a data set of 27 calibration positions of the robot. A detailed description for such a procedure can be found in [13].

## 3  Results

### 3.1  System performance

The accuracy of the current system allows to estimate the pointing target to an accuracy of $1 \pm 0.4$ cm (average over $N = 7$ objects at randomly chosen locations in the workspace) in a workspace area of 50x50 cm. In our current environment, this is sufficient to pick and place any of the seven defined target objects at any location in the workspace. This accuracy can only be achieved if we use the object recognition module described in sec. 2. The output of the pointing direction module approximates the target location with an considerably lower accuracy of $3.6 \pm 1.6$ cm.

### 3.2  Image segmentation

The problem to evaluate these preprocessing steps has been discussed previously [7], especially the relation of specifity and sensitivity of the network for the given task. As the pointing recognition is based on a subframe centered on the hand center, it is very sensitive to deviations from this center so that a good localization accuracy

is even more important than the classification rate. The localization accuracy is calculated by measuring the pixel distance between the centers determined manually on the original image and as the center of mass in the image obtained after application of the neural network. Table 1 provides quantitative results.

On the whole, the two-step cascade of LLM networks yields for *399 out of 400 images* an activity image precisely centered on the human hand. Only in one image, the first LLM net missed the hand completely, due to a second hand in the image that could be clearly seen in this view. This image was excluded from further processing and from the evaluation of the localization accuracy.

| | Camera A | | Camera B | |
|---|---|---|---|---|
| | Pixel deviation | NRMSE | Pixel deviation | NRMSE |
| Person A | $0.8 \pm 1.2$ | $0.03 \pm 0.06$ | $0.8 \pm 2.2$ | $0.03 \pm 0.09$ |
| Person B | $1.3 \pm 1.4$ | $0.06 \pm 0.11$ | $2.2 \pm 2.8$ | $0.11 \pm 0.21$ |

Table 1: *Estimation error of the hand localization on the test set. Absolute error in pixels and normalized error for both persons and both camera images.*

## 3.3  Recognition performance

One major problem in recognizing human pointing gestures is the variability of these gestures and their measurement for the acquisition of reliable training information. Different persons follow different strategies where and how to point (fig. 2 (center) and (right)). Therefore, we calculate this information indirectly. The person is told to point at a certain grid position with known world coordinates. From the camera images we extract the pixel positions of the hand center and map them to world coordinates using the PSOM net (P1 in fig. 1). Given these coordinates the angles of the intended pointing vector with the basis vectors of the world coordinate system can be calculated trigonometrically. These angles form the target vector for the supervised training of a LLM network (D in fig. 1).

After training, the output of the net is used to calculate the point where the pointing vector intersects the table surface. For evaluation of the network performance we measure the Euclidian distance between this point and the actual grid point where the person intended to point at. Fig. 3 *(left)* shows the mean euclidean error MEE of the estimated target position as a function of the number of learning steps. The error on the training set can be considerably reduced, whereas on the test set the improvement stagnates after some 500 training steps. If we perform even more training steps the performance might actually suffer from overfitting. The graph compares training and test results achieved on images obtained by two different ways of determining the hand center. The "manual" curves show the performance that can be achieved if the Gabor masks are manually centered on the hand. For the "neuronal" curves, the center of mass calculated in the fine-segmented and post-processed subframe was used. This allows us to study the influence of the error of the segmentation and localization steps on the pointing recognition. This influence is rather small. The MEE increases from 17 mm for the optimal method to 19 mm for the neural method, which is hardly visible in practice.

The curves in fig. 3 *(center)* are obtained if we apply the networks to images of another person. The MEE is considerably larger but a detailed analysis' shows that part of this deviation is due to systematic differences in the pointing strategy as shown in fig. 2 *(right)*. Over a wide range, the number of nodes used for the LLM network has only minor influence on the performance. While obviously the performance on the training set can be arbitrarily improved by spending more nodes, the differences in the MEE on the test set are negligible in a range of 5 to 15 nodes. Using more nodes is problematic as the training data consists of 50 examples only. If not indicated otherwise, we use LLM networks with 10 nodes. Further results,

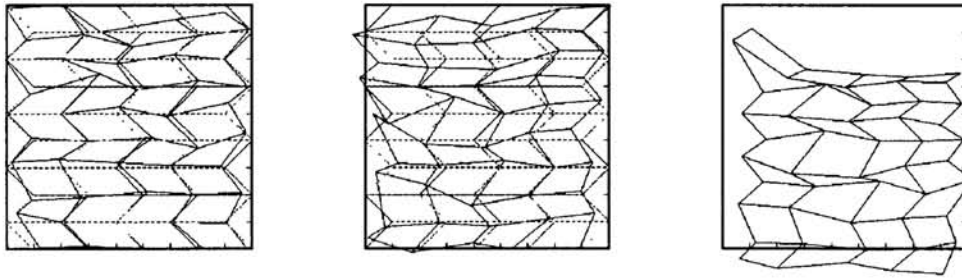

Fig. 2: *The table grid points can be reconstructed according to the network output. The target grid is dotted. Reconstruction of training grid (left) and test grid (center) for one person, and of the test grid for another person (right).*

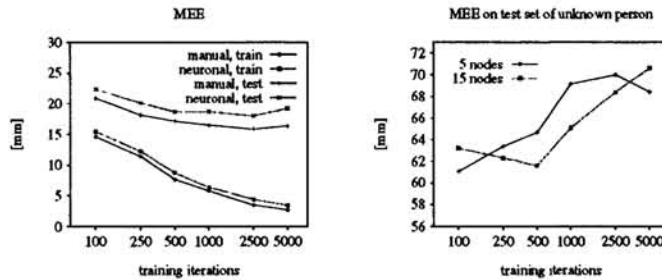

Fig. 3: *The euclidean error of estimated target point calculated using the network output depends on the preprocessing (left), and the person (center).*

comparing the pointing recognition based on only one of the camera images, indicate that the method works better if the camera takes a lateral view rather than a frontal view. All evaluations were done for both persons. The performance was always very similar.

## 4   Discussion

While we begin to understand many properties of neural networks at the single network level, our insight into principled ways of how to build *neural systems* is still rather limited. Due to the complexity of this task, theoretical progress is (and probably will continue to be) very slow. What we can do in the mean time, however, is to experiment with different design strategies for neural systems and try to "evolve" useful approaches by carefully chosen case studies.

The current work is an effort along these lines. It is focused on a challenging, practically important vision task with a number of generic features that are shared with vision tasks for which biological vision systems were evolved.

One important issue is how to achieve robustness at the different processing levels of the system. There are only very limited possibilities to study this issue in simulations, since practically nothing is known about the statistical properties of the various sources of error that occur when dealing with real world data. Thus, a real implementation that works with actual data is practically the only way to study the robustness issue in a realistic fashion. Therefore, the demonstrated integration of several functional modules that we had developed previously in more restricted settings [7, 6] was a non-trivial test of the feasability of having these functions cooperate in a larger, modular system. It also gives confidence that the scaling problem can be dealt with successfully if we apply modular neural nets.

A related and equally important issue was the use of a processing strategy in which earlier processing stages *incrementally restrict the search space* for the subsequent stages. Thus, the responsibility for achieving the goal is not centralized in any single module and subsequent modules have always the chance to compensate for limited errors of earlier stages. This appears to be a generally useful strategy for achieving

robustness and for cutting computational costs that is related to the use of "focal attention", which is clearly an important element of many biological vision systems.

A third important point is the extensive use of learning to build the essential constituent functions of the system from data examples. We are not yet able to train the assembled system as a whole. Instead, different modules are trained separately and are integrated only later. Still, the experience gained with assembling a complex system via this "engineering-type" of approach will be extremely valuable for gradually developing the capability of crafting larger functional building blocks by learning methods.

We conclude that carefully designed experiments with modular neural systems that are based on the use of real world data and that focus on similar tasks for which also biological neural systems were evolved can make a significant contribution in tackling the challenge that lies ahead of us: to develop a reliable technology for the construction of *large-scale artificial neural systems* that can solve complex tasks in real world environments.

### Acknowledgements

We want to thank Th. Wengerek (robot control), J. Walter (PSOM implementation), and P. Ziemeck (image acquisition software). This work was supported by BMFT Grant No. ITN9104AO.

## Footnotes

[1]In analogy to the sea eagle who watches its prey from high above, shoots down to grab the prey, and then flies to a safe place to feed, we nicknamed our system "SEE-EAGLE".

[2]Development by Prof. Pfeiffer, TU Munich

## References

[1] T. J. Darell and A. P. Pentland. Classifying hand gestures with a view-based distributed representation. In J. D. Cowan, G. Tesauro, and J. Alspector, editors, *Neural Information Processing Systems 6*, pages 945–952. Morgan Kaufman, 1994.

[2] J. Davis and M. Shah. Recognizing hand gestures. In J.-O. Eklundh, editor, *Computer Vision — ECCV '94*, volume 800 of *Lecture Notes in Computer Science*, pages 331–340. Springer-Verlag, Berlin Heidelberg New York, 1994.

[3] R.A. Jacobs, M.I. Jordan, S.J. Nowlan, and G.E. Hinton. Adaptive mixtures of local experts. *Neural Computation*, 3:79–87, 1991.

[4] M.I. Jordan and R.A. Jacobs. Hierarchical mixtures of experts and the EM algorithm. *Neural Computation*, 6(2):181–214, 1994.

[5] F. Kummert, E. Littmann, A. Meyering, S. Posch, H. Ritter, and G. Sagerer. A hybrid approach to signal interpretation using neural and semantic networks. In *Mustererkennung 1993*, pages 245–252. Springer, 1993.

[6] E. Littmann, A. Drees, and H. Ritter. Neural recognition of human pointing gestures in real images. Submitted to *Neural Processing Letters*, 1996.

[7] E. Littmann and H. Ritter. Neural and statistical methods for adaptive color segmentation — a comparison. In G. Sagerer, S. Posch, and F. Kummert, editors, *Mustererkennung 1995*, pages 84–93. Springer-Verlag, Heidelberg, 1995.

[8] C. Maggioni. A novel device for using the hand as a human-computer interface. In *Proceedings HCI'93 — Human Control Interface*, Loughborough, Great Britain, 1993.

[9] A. Meyering and H. Ritter. Learning 3D shape perception with local linear maps. In *Proc. of the IJCNN*, volume IV, pages 432–436, Baltimore, MD, 1992.

[10] Steven J. Nowlan and John C. Platt. A convolutional neural network hand tracker. In *Neural Information Processing Systems 7*. Morgan Kaufman Publishers, 1995.

[11] H. Ritter. Parametrized self-organizing maps for vision learning tasks. In P. Morasso, editor, *ICANN '94*. Springer-Verlag, Berlin Heidelberg New York, 1994.

[12] K. Väänänen and K. Böhm. Gesture driven interaction as a human factor in virtual environments – an approach with neural networks. In R. Earnshaw, M. Gigante, and H. Jones, editors, *Virtual reality systems*, pages 93–106. Academic Press, 1993.

[13] J. Walter and H. Ritter. Rapid learning with parametrized self-organizing maps. *Neural Computing*, 1995. Submitted.

[14] T. G. Zimmermann, J. Lanier, C. Blanchard, S. Bryson, and Y. Harvill. A hand gesture interface device. In *Proc. CHI+GI*, pages 189–192, 1987.
